# LEARNING ON A GENERAL NETWORK

Amir F. Atiya
Department of Electrical Engineering
California Institute of Technology
Ca 91125

## Abstract

This paper generalizes the backpropagation method to a general network containing feedback connections. The network model considered consists of interconnected groups of neurons, where each group could be fully interconnected (it could have feedback connections, with possibly asymmetric weights), but no loops between the groups are allowed. A stochastic descent algorithm is applied, under a certain inequality constraint on each intra-group weight matrix which ensures for the network to possess a unique equilibrium state for every input.

## Introduction

It has been shown in the last few years that large networks of interconnected "neuron"-like elements are quite suitable for performing a variety of computational and pattern recognition tasks. One of the well-known neural network models is the backpropagation model [1]-[4]. It is an elegant way for teaching a layered feedforward network by a set of given input/output examples. Neural network models having feedback connections, on the other hand, have also been devised (for example the Hopfield network [5]), and are shown to be quite successful in performing some computational tasks. It is important, though, to have a method for learning by examples for a feedback network, since this is a general way of design, and thus one can avoid using an ad hoc design method for each different computational task. The existence of feedback is expected to improve the computational abilities of a given network. This is because in feedback networks the state iterates until a stable state is reached. Thus processing is performed on several steps or recursions. This, in general allows more processing abilities than the "single step" feedforward case (note also the fact that a feedforward network is a special case of a feedback network). Therefore, in this work we consider the problem of developing a general learning algorithm for feedback networks.

In developing a learning algorithm for feedback networks, one has to pay attention to the following (see Fig. 1 for an example of a configuration of a feedback network). The state of the network evolves in time until it goes to equilibrium, or possibly other types of behavior such as periodic or chaotic motion could occur. However, we are interested in having a steady and and fixed output for every input applied to the network. Therefore, we have the following two important requirements for the network. Beginning in any initial condition, the state should ultimately go to equilibrium. The other requirement is that we have to have a unique

equilibrium state. It is in fact that equilibrium state that determines the final output. The objective of the learning algorithm is to adjust the parameters (weights) of the network in small steps, so as to move the unique equilibrium state in a way that will result finally in an output as close as possible to the required one (for each given input). The existence of more than one equilibrium state for a given input causes the following problems. In some iterations one might be updating the weights so as to move one of the equilibrium states in a sought direction, while in other iterations (especially with different input examples) a different equilibrium state is moved. Another important point is that when implementing the network (after the completion of learning), for a fixed input there can be more than one possible output. Independently, other work appeared recently on training a feedback network [6],[7],[8]. Learning algorithms were developed, but solving the problem of ensuring a unique equilibrium was not considered. This problem is addressed in this paper and an appropriate network and a learning algorithm are proposed.

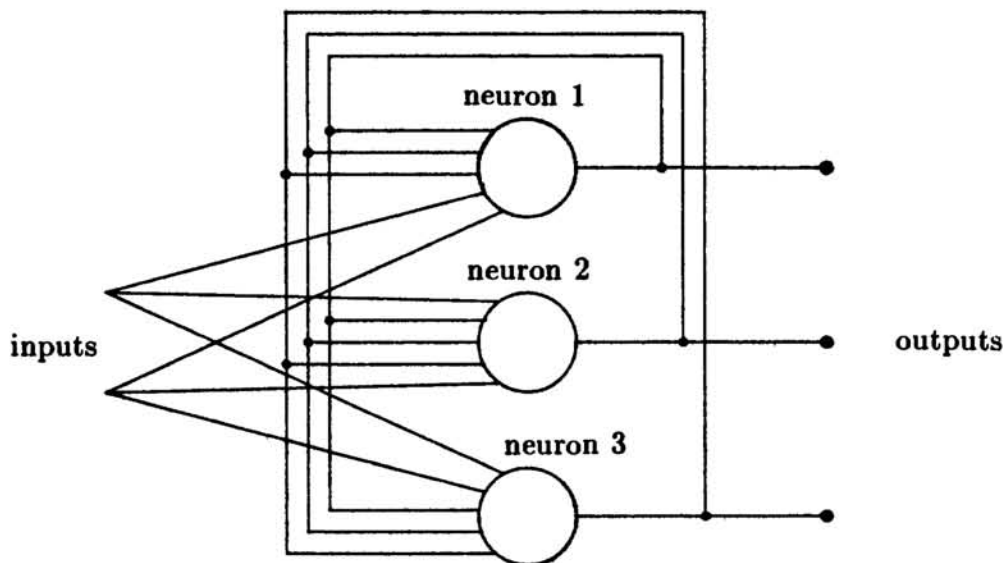

Fig. 1
A recurrent network

### The Feedback Network

Consider a group of $n$ neurons which could be fully inter-connected (see Fig. 1 for an example). The weight matrix $\mathbf{W}$ can be asymmetric (as opposed to the Hopfield network). The inputs are also weighted before entering into the network (let $\mathbf{V}$ be the weight matrix). Let $\mathbf{x}$ and $\mathbf{y}$ be the input and output vectors respectively. In our model $\mathbf{y}$ is governed by the following set of differential equations, proposed by Hopfield [5]:

$$\tau\frac{d\mathbf{u}}{dt} = \mathbf{W}f(\mathbf{u}) - \mathbf{u} + \mathbf{V}\mathbf{x}, \qquad \mathbf{y} = \mathbf{f}(\mathbf{u}) \qquad (1)$$

where $\mathbf{f(u)} = (f(u_1), ..., f(u_n))^T$, $T$ denotes the transpose operator, $f$ is a bounded and differentiable function, and $\tau$ is a positive constant.

For a given input, we would like the network after a short transient period to give a steady and fixed output, no matter what the initial network state was. This means that beginning any initial condition, the state is to be attracted towards a unique equilibrium. This leads to looking for a condition on the matrix $\mathbf{W}$.

*Theorem:* A network (not necessarily symmetric) satisfying

$$\sum_i \sum_j w_{ij}^2 < 1/\max(f')^2,$$

exhibits no other behavior except going to a unique equilibrium for a given input.

*Proof* : Let $\mathbf{u}_1(t)$ and $\mathbf{u}_2(t)$ be two solutions of (1). Let

$$J(t) = \|\mathbf{u}_1(t) - \mathbf{u}_2(t)\|^2$$

where $\| \ \|$ is the two-norm. Differentiating $J$ with respect to time, one obtains

$$\frac{dJ(t)}{dt} = 2(\mathbf{u}_1(t) - \mathbf{u}_2(t))^T \left( \frac{d\mathbf{u}_1(t)}{dt} - \frac{d\mathbf{u}_2(t)}{dt} \right).$$

Using (1) , the expression becomes

$$\frac{dJ(t)}{dt} = -\frac{2}{\tau}\|\mathbf{u}_1(t) - \mathbf{u}_2(t)\|^2 + \frac{2}{\tau}(\mathbf{u}_1(t) - \mathbf{u}_2(t))^T \mathbf{W}\big[\mathbf{f}(\mathbf{u}_1(t)) - \mathbf{f}(\mathbf{u}_2(t))\big].$$

Using Schwarz's Inequality, we obtain

$$\frac{dJ(t)}{dt} \leq -\frac{2}{\tau}\|\mathbf{u}_1(t) - \mathbf{u}_2(t)\|^2 + \frac{2}{\tau}\|\mathbf{u}_1(t) - \mathbf{u}_2(t)\| \cdot \|\mathbf{W}\big[\mathbf{f}(\mathbf{u}_1(t)) - \mathbf{f}(\mathbf{u}_2(t))\big]\|.$$

Again, by Schwarz's Inequality,

$$\mathbf{w}_i\big[\mathbf{f}(\mathbf{u}_1(t)) - \mathbf{f}(\mathbf{u}_2(t))\big] \leq \|\mathbf{w}_i\| \cdot \|\mathbf{f}(\mathbf{u}_1(t)) - \mathbf{f}(\mathbf{u}_2(t))\|, \qquad i = 1, ..., n \qquad (2)$$

where $\mathbf{w}_i$ denotes the $i^{th}$ row of $\mathbf{W}$. Using the mean value theorem, we get

$$\|\mathbf{f}(\mathbf{u}_1(t)) - \mathbf{f}(\mathbf{u}_2(t))\| \leq (\max|f'|)\|\mathbf{u}_1(t) - \mathbf{u}_2(t)\|. \qquad (3)$$

Using (2),(3), and the expression for $J(t)$, we get

$$\frac{dJ(t)}{dt} \leq -\alpha J(t) \qquad (4)$$

where

$$\alpha = \frac{2}{\tau} - \frac{2}{\tau}(\max|f'|)\sqrt{\sum_i \sum_j w_{ij}^2}.$$

By hypothesis of the Theorem, $\alpha$ is strictly positive. Multiplying both sides of (4) by $exp(\alpha t)$, the inequality

$$\frac{d}{dt}(J(t)e^{\alpha t}) \leq 0$$

results, from which we obtain

$$J(t) \leq J(0)e^{-\alpha t}.$$

From that and from the fact that $J$ is non-negative, it follows that $J(t)$ goes to zero as $t \to \infty$. Therefore, any two solutions corresponding to any two initial conditions ultimately approach each other. To show that this asymptotic solution is in fact an equilibrium, one simply takes $\mathbf{u}_2(t) = \mathbf{u}_1(t+T)$, where $T$ is a constant, and applies the above argument (that $J(t) \to 0$ as $t \to \infty$), and hence $\mathbf{u}_1(t+T) \to \mathbf{u}_1(t)$ as $t \to \infty$ for any $T$, and this completes the proof.

For example, if the function $f$ is of the following widely used sigmoid-shaped form,

$$f(u) = \frac{1}{1+e^{-u}},$$

then the sum of the square of the weights should be less than 16. Note that for any function $f$, scaling does not have an effect on the overall results. We have to work in our updating scheme subject to the constraint given in the Theorem. In many cases where a large network is necessary, this constraint might be too restrictive. Therefore we propose a general network, which is explained in the next Section.

**The General Network**

We propose the following network (for an example refer to Fig. 2). The neurons are partitioned into several groups. Within each group there are no restrictions on the connections and therefore the group could be fully interconnected (i.e. it could have feedback connections). The groups are connected to each other, but in a way that there are no loops. The inputs to the whole network can be connected to the inputs of any of the groups (each input can have several connections to several groups). The outputs of the whole network are taken to be the outputs (or part of the outputs) of a certain group, say group $f$. The constraint given in the Theorem is applied on each intra-group weight matrix separately. Let $(\mathbf{q}^a, \mathbf{s}^a)$, $a = 1, ..., N$ be the input/output vector pairs of the function to be implemented. We would like to minimize the sum of the square error, given by

$$E = \sum_{a=1}^{N} e_a$$

where

$$e_a = \sum_{i=1}^{M}(y_i^f - s_i^a)^2,$$

and $\mathbf{y}^f$ is the output vector of group $f$ upon giving input $\mathbf{q}^a$, and $M$ is the dimension of vector $\mathbf{s}^a$. The learning process is performed by feeding the input examples $\mathbf{q}^a$ sequentially to the network, each time updating the weights in an attempt to minimize the error.

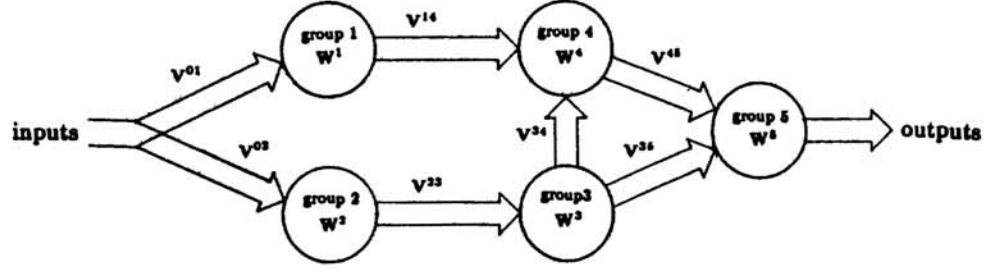

Fig. 2
An example of a general network
(each group represents a recurrent network)

Now, consider a single group $l$. Let $\mathbf{W}^l$ be the intra-group weight matrix of group $l$, $\mathbf{V}^{rl}$ be the matrix of weights between the outputs of group $r$ and the inputs of group $l$, and $\mathbf{y}^l$ be the output vector of group $l$. Let the respective elements be $w_{ij}^l$, $v_{ij}^{rl}$, and $y_i^l$. Furthermore, let $n_l$ be the number of neurons of group $l$. Assume that the time constant $\tau$ is sufficiently small so as to allow the network to settle quickly to the equilibrium state, which is given by the solution of the equation

$$\mathbf{y}^l = \mathbf{f}(\mathbf{W}^l\mathbf{y}^l + \sum_{r \epsilon A_l} \mathbf{V}^{rl}\mathbf{y}^r). \qquad (5)$$

where $A_l$ is the set of the indices of the groups whose outputs are connected to the inputs of group $l$. We would like each iteration to update the weight matrices $\mathbf{W}^l$ and $\mathbf{V}^{rl}$ so as to move the equilibrium in a direction to decrease the error. We need therefore to know the change in the error produced by a small change in the weight matrices. Let $\frac{\partial e_a}{\partial \mathbf{W}^l}$, and $\frac{\partial e_a}{\partial \mathbf{V}^{rl}}$ denote the matrices whose $(i,j)^{th}$ element are $\frac{\partial e_a}{\partial w_{ij}^l}$, and $\frac{\partial e_a}{\partial v_{ij}^{rl}}$ respectively. Let $\frac{\partial e_a}{\partial \mathbf{y}^l}$ be the column vector whose $i^{th}$ element is $\frac{\partial e_a}{\partial y_i^l}$. We obtain the following relations:

$$\frac{\partial e_a}{\partial \mathbf{W}^l} = \left[\mathbf{\Lambda}^l - (\mathbf{W}^l)^T\right]^{-1} \frac{\partial e_a}{\partial \mathbf{y}^l}(\mathbf{y}^l)^T,$$

$$\frac{\partial e_a}{\partial \mathbf{V}^{rl}} = \left[\mathbf{\Lambda}^l - (\mathbf{W}^l)^T\right]^{-1} \frac{\partial e_a}{\partial \mathbf{y}^l}(\mathbf{y}^r)^T,$$

where $\mathbf{\Lambda}^l$ is the diagonal matrix whose $i^{th}$ diagonal element is $1/f'(\sum_k w_{ik}^l y_k^l + \sum_r \sum_k v_{ik}^{rl} y_k^r)$ for a derivation refer to Appendix). The vector $\frac{\partial e_a}{\partial \mathbf{y}^l}$ associated with group $l$ can be obtained in terms of the vectors $\frac{\partial e_a}{\partial \mathbf{y}^j}$, $j \epsilon B_l$, where $B_l$ is the set of the indices of the groups whose inputs are connected to the outputs of group $l$. We get (refer to Appendix)

$$\frac{\partial e_a}{\partial \mathbf{y}^l} = \sum_{j \epsilon B_l} (\mathbf{V}^{lj})^T [\mathbf{\Lambda}^j - (\mathbf{W}^j)^T]^{-1} \frac{\partial e_a}{\partial \mathbf{y}^j}. \qquad (6)$$

The matrix $\mathbf{\Lambda}^l - (\mathbf{W}^l)^T$ for any group $l$ can never be singular, so we will not face any problem in the updating process. To prove that, let $\mathbf{z}$ be a vector satisfying

$$[\mathbf{\Lambda}^l - (\mathbf{W}^l)^T]\mathbf{z} = 0.$$

We can write

$$z_i/\max|f'| \leq \sum_k w^l_{ki} z_k, \qquad i = 1, ..., n_l$$

where $z_i$ is the $i^{th}$ element of $\mathbf{z}$. Using Schwarz's Inequality, we obtain

$$|z_i|/\max|f'| \leq \sqrt{\sum_k z_k^2} \sqrt{\sum_k (w^l_{ki})^2}, \qquad i = 1, ..., n_l$$

Squaring both sides and adding the inequalities for $i = 1, ..., n_l$, we get

$$\sum_i z_i^2 \leq \max(f')^2 (\sum_k z_k^2) \sum_i \sum_k (w^l_{ki})^2. \qquad (7)$$

Since the condition

$$\sum_i \sum_k (w^l_{ik})^2 < 1/\max(f')^2),$$

is enforced, it follows that (7) cannot be satisfied unless $\mathbf{z}$ is the zero vector. Thus, the matrix $\mathbf{\Lambda}^l - (\mathbf{W}^l)^T$ cannot be singular.

For each iteration we begin by updating the weights of group $f$ (the group containing the final outputs). For that group $\frac{\partial e_a}{\partial \mathbf{y}}$ equals simply $2(y_1^f - s_1, ..., y_M^f - s_M, 0, ..., 0)^T$. Then we move backwards to the groups connected to that group and obtain their corresponding $\frac{\partial e_a}{\partial \mathbf{y}}$ vectors using (6), update the weights, and proceed in the same manner until we complete updating all the groups. Updating the weights is performed using the following stochastic descent algorithm for each group,

$$\Delta \mathbf{W} = -\alpha_1 \frac{\partial e_a}{\partial \mathbf{W}} + \alpha_2 e_a \mathbf{R},$$

$$\Delta \mathbf{V} = -\alpha_3 \frac{\partial e_a}{\partial \mathbf{V}} + \alpha_4 e_a \mathbf{R},$$

where $\mathbf{R}$ is a noise matrix whose elements are characterized by independent zero-mean unity-variance Gaussian densities, and the $\alpha$'s are parameters. The purpose of adding noise is to allow escaping local minima if one gets stuck in any of them. Note that the control parameter is taken to be $e_a$. Hence the variance of the added noise tends to decrease the more we approach the ideal zero-error solution. This makes sense because for a large error, i.e. for an unsatisfactory solution, it pays more to add noise to the weight matrices in order to escape local minima. On the other hand, if the error is small, then we are possibly near the global minimum or to an acceptable solution, and hence we do not want too much noise in order not to be thrown out of that basin. Note that once we reach the ideal zero-error solution the added noise as well as the gradient of $e_a$ become zero for all $a$ and hence the increments of the weight matrices become zero. If after a certain iteration $\mathbf{W}$ happens to violate the constraint $\sum_{ij} w_{ij}^2 \leq constant < 1/max(f')^2$, then its elements are scaled so as to project it back onto the surface of the hypershere.

## Implementation Example

A pattern recognition example is considered. Fig. 3 shows a set of two-dimensional training patterns from three classes. It is required to design a neural network recognizer with

three output neurons. Each of the neurons should be on if a sample of the corresponding class is presented, and off otherwise, i.e. we would like to design a "winner-take-all" network. A single-layer three neuron feedback network is implemented. We obtained 3.3% error. Performing the same experiment on a feedforward single-layer network with three neurons, we obtained 20% error. For satisfactory results, a feedforward network should be two-layer. With one neuron in the first layer and three in the second layer, we got 36.7% error. Finally, with two neurons in the first layer and three in the second layer, we got a match with the feedback case, with 3.3% error.

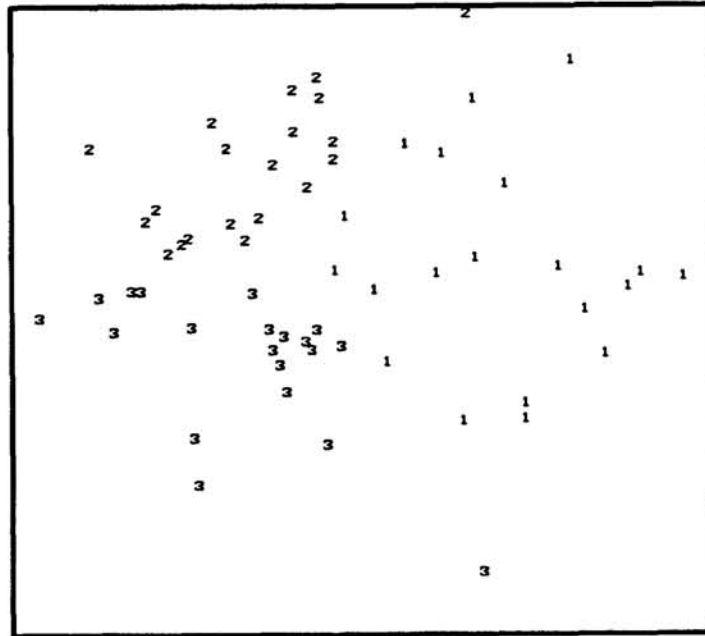

Fig. 3
A pattern recognition example

## Conclusion

A way to extend the backpropagation method to feedback networks has been proposed. A condition on the weight matrix is obtained, to insure having only one fixed point, so as to prevent having more than one possible output for a fixed input. A general structure for networks is presented, in which the network consists of a number of feedback groups connected to each other in a feedforward manner. A stochastic descent rule is used to update the weights. The method is applied to a pattern recognition example. With a single-layer feedback network it obtained good results. On the other hand, the feedforward backpropagation method achieved good resuls only for the case of more than one layer, hence also with a larger number of neurons than the feedback case.

## Acknowledgement

The author would like to gratefully acknowledge Dr. Y. Abu-Mostafa for the useful discussions. This work is supported by Air Force Office of Scientific Research under Grant AFOSR-86-0296.

## Appendix

Differentiating (5), one obtains

$$\frac{\partial y_j^l}{\partial w_{kp}^l} = f'(z_j^l)(\sum_m w_{jm}^l \frac{\partial y_m^l}{\partial w_{kp}^l} + y_p^l \delta_{jk}), \qquad k, p = 1, ..., n_l$$

where

$$\delta_{jk} = \begin{cases} 1 & \text{if } j = k \\ 0 & \text{otherwise,} \end{cases}$$

and

$$z_j^l = \sum_m w_{jm}^l y_m^l + \sum_{r \epsilon A_l} \sum_m v_{jm}^{rl} y_m^r.$$

We can write

$$\frac{\partial \mathbf{y}^l}{\partial w_{kp}^l} = (\mathbf{\Lambda}^l - \mathbf{W}^l)^{-1} \mathbf{b}^{kp} \qquad (A-1)$$

where $\mathbf{b}^{kp}$ is the $n_l$-dimensional vector whose $i^{th}$ component is given by

$$b_i^{kp} = \begin{cases} y_p^l & \text{if } i = k \\ 0 & \text{otherwise.} \end{cases}$$

By the chain rule,

$$\frac{\partial e_a}{\partial w_{kp}^l} = \sum_j \frac{\partial e_a}{\partial y_j^l} \frac{\partial y_j^l}{\partial w_{kp}^l},$$

which, upon substituting from $(A-1)$, can be put in the form $y_p^l \mathbf{g}_k^T \frac{\partial e_a}{\partial \mathbf{y}^l}$, where $\mathbf{g}_k$ is the $k^{th}$ column of $(\mathbf{\Lambda}^l - \mathbf{W}^l)^{-1}$. Finally, we obtain the required expression, which is

$$\frac{\partial e_a}{\partial \mathbf{W}^l} = \left[\mathbf{\Lambda}^l - (\mathbf{W}^l)^T\right]^{-1} \frac{\partial e_a}{\partial \mathbf{y}^l} (\mathbf{y}^l)^T.$$

Regarding $\frac{\partial e_a}{\partial \mathbf{V}^{rl}}$, it is obtained by differentiating (5) with respect to $v_{kp}^{rl}$. We get similarly

$$\frac{\partial \mathbf{y}^l}{\partial v_{kp}^{rl}} = (\mathbf{\Lambda}^l - \mathbf{W}^l)^{-1} \mathbf{c}^{kp}$$

where $\mathbf{c}^{kp}$ is the $n_l$-dimensional vector whose $i^{th}$ component is given by

$$c_i^{kp} = \begin{cases} y_p^r & \text{if } i = k \\ 0 & \text{otherwise.} \end{cases}$$

A derivation very similar to the case of $\frac{\partial e_a}{\partial \mathbf{W}^l}$ results in the following required expression:

$$\frac{\partial e_a}{\partial \mathbf{V}^{rl}} = \left[\mathbf{\Lambda}^l - (\mathbf{W}^l)^T\right]^{-1} \frac{\partial e_a}{\partial \mathbf{y}^l} (\mathbf{y}^r)^T.$$

Now, finally consider $\frac{\partial e_a}{\partial \mathbf{y}^l}$. Let $\frac{\partial \mathbf{y}^j}{\partial \mathbf{y}^l}$, $j\epsilon B_l$ be the matrix whose $(k,p)^{th}$ element is $\frac{\partial y_k^j}{\partial y_p^l}$. The elements of $\frac{\partial \mathbf{y}^j}{\partial \mathbf{y}^l}$ can be obtained by differentiating the equation for the fixed point for group $j$, as follows,

$$\frac{\partial y_k^j}{\partial y_p^l} = f'(z_k^j)(v_{kp}^{lj} + \sum_m w_{km}^j \frac{\partial y_m^j}{\partial y_p^l}).$$

Hence,

$$\frac{\partial \mathbf{y}^j}{\partial \mathbf{y}^l} = (\mathbf{\Lambda}^j - \mathbf{W}^j)^{-1} \mathbf{V}^{lj}. \qquad (A-2)$$

Using the chain rule, one can write

$$\frac{\partial e_a}{\partial \mathbf{y}^l} = \sum_{j\epsilon B_l} \left(\frac{\partial \mathbf{y}^j}{\partial \mathbf{y}^l}\right)^T \frac{\partial e_a}{\partial \mathbf{y}^j}.$$

We substitute from $(A-2)$ into the previous equation to complete the derivation by obtaining

$$\frac{\partial e_a}{\partial \mathbf{y}^l} = \sum_{j\epsilon B_l} (\mathbf{V}^{lj})^T \left[(\mathbf{\Lambda}^j - (\mathbf{W}^j)^T)\right]^{-1} \frac{\partial e_a}{\partial \mathbf{y}^j}.$$

### References

[1] P. Werbos, "Beyond regression: New tools for prediction and analysis in behavioral sciences", Harvard University dissertation, 1974.

[2] D. Parker, "Learning logic", MIT Tech Report TR-47, Center for Computational Research in Economics and Management Science, 1985.

[3] Y. Le Cun, "A learning scheme for asymmetric threshold network", Proceedings of Cognitiva, Paris, June 1985.

[4] D. Rumelhart, G.Hinton, and R. Williams, "Learning internal representations by error propagation", in D. Rumelhart, J. McLelland and the PDP research group (Eds.), *Parallel distributed processing: Explorations in the microstructure of cognition*, Vol. 1, MIT Press, Cambridge, MA, 1986.

[5] J. Hopfield, "Neurons with graded response have collective computational properties like those of two-state neurons", Proc. Natl. Acad. Sci. USA, May 1984.

[6] L. Almeida, "A learning rule for asynchronous perceptrons with feedback in a combinatorial environment", Proc. of the First Int. Annual Conf. on Neural Networks, San Diego, June 1987.

[7] R. Rohwer, and B. Forrest, "Training time-dependence in neural networks", Proc. of the First Int. Annual Conf. on Neural Networks, San Diego, June 1987.

[8] F. Pineda, "Generalization of back-propagation to recurrent neural networks", Phys. Rev. Lett., vol. 59, no. 19, 9 Nov. 1987.
